# Boosted Dyadic Kernel Discriminants

**Baback Moghaddam**
Mitsubishi Electric Research Laboratory
201 Broadway
Cambridge MA 02139 USA
*baback@merl.com*

**Gregory Shakhnarovich**
MIT AI Laboratory
200 Technology Square
Cambridge MA 02139 USA
*gregory@ai.mit.edu*

## Abstract

We introduce a novel learning algorithm for binary classification with hyperplane discriminants based on pairs of training points from opposite classes (*dyadic hypercuts*). This algorithm is further extended to nonlinear discriminants using kernel functions satisfying Mercer's conditions. An ensemble of simple dyadic hypercuts is learned incrementally by means of a confidence-rated version of AdaBoost, which provides a sound strategy for searching through the finite set of hypercut hypotheses. In experiments with real-world datasets from the UCI repository, the generalization performance of the hypercut classifiers was found to be comparable to that of SVMs and k-NN classifiers. Furthermore, the computational cost of classification (at run time) was found to be similar to, or better than, that of SVM. Similarly to SVMs, boosted dyadic kernel discriminants tend to maximize the margin (via AdaBoost). In contrast to SVMs, however, we offer an on-line and incremental learning machine for building kernel discriminants whose complexity (number of kernel evaluations) can be directly controlled (traded off for accuracy).

## 1 Introduction

This paper introduces a novel algorithm for learning complex binary classifiers by superposition of simpler hyperplane-type discriminants. In this algorithm, each of the simple discriminants is based on the projection of a test point onto a vector joining a *dyad*, defined as a pair of training data points with opposite labels. The learning algorithm itself is based on a real-valued variant of AdaBoost [7], and the hyperplane classifiers use kernels of the type used, e.g., by support vector machines (SVMs) [9] for mapping linearly non-separable problems to high-dimensional feature spaces.

When the concept class consists of linear discriminants (hyperplanes), this amounts to using a hyperplane orthogonal to the vector connecting the point in a dyad. We shall refer to such a classifier as a *hypercut*. By applying the same notion of linear hypercuts to a nonlinearly transformed feature space obtained by Mercer-type kernels [3], we are able to implement nonlinear kernel discriminants similar in form to SVMs.

In each iteration of AdaBoost, the space of all dyadic hypercuts is searched. It can be easily shown that this hypothesis space spans the subspace of the data and that it must include the optimal hyperplane discriminant. This notion is readily extended to non-linear classifiers obtained by kernel transformations, by noting that in the *feature* space, the optimal discriminant resides in the span of the *transformed* data. Therefore, for both linear and nonlinear classification, searching the space of dyadic hypercuts forms an efficient strategy for exploring the space of all hypotheses.

## 1.1 Related work

The most general framework to consider is the theory of *potential functions* for pattern classification [1] in which *potential fields*[1] of the form

$$H(\mathbf{x}) = \sum_i \alpha_i y_i K(\mathbf{x}, \mathbf{x}_i) \tag{1}$$

are thresholded to predict classification labels, $\hat{y} = \text{sign}(H(\mathbf{x}))$. In a probabilistic kernel regression framework recently proposed in [5], the coefficients $\alpha$ that minimize the classification error are obtained by maximizing

$$J(\alpha) = -\frac{1}{2} \sum_{i,j} \alpha_i \alpha_j y_i y_j K(\mathbf{x_i}, \mathbf{x_j}) + \sum_i F(\alpha_i), \tag{2}$$

where the potential function $F$ is concave and continuous (corresponding to positive semi-definite kernels). This framework subsumes SVMs, which correspond to the simplest case $F(\alpha) = \alpha$. Generalized linear models [6] can also be shown to be members of this class by considering logistic regression where $F(\alpha)$ becomes the binary entropy function and $K$ is related to the covariance function of a Gaussian process classifier for the GLM's intermediate variables.

In this paper we propose and design classifiers with dyadic discriminants, which have potential functions of the form

$$H(\mathbf{x}) = \sum_t \alpha_t K(\mathbf{x}, \mathbf{x}_t^p) - \alpha_t K(\mathbf{x}, \mathbf{x}_t^n), \tag{3}$$

where $\mathbf{x}^p$ and $\mathbf{x}^n$ are positively and negatively labeled data, respectively. The coefficients $\alpha_t$ are determined not by minimizing a convex quadratic function $J(\alpha)$ but rather by selecting an optimal classifier in the $t$-th iteration of AdaBoost. Thus the potential function is constrained to the form of a weighted sum of dyadic hypercuts, or *differences* of kernel functions. Another way to view this is to think of a pair of opposite – polarity "basis vectors" sharing the same coefficient $\alpha_t$.

The most closely related potential function technique to ours is that of SVMs [9], where the classification margin (and thus the bound on generalization) is maximized by a simultaneous optimization with respect to all of the training points. However, there are important differences between SVMs and our iterative hypercut algorithm. In each step of the boosting process, we do not maximize the margin of the resulting strong classifier directly, which makes for a much simpler optimization task. Meanwhile, we are assured that with AdaBoost we tend to maximize (although in an asymptotic sense) the margin of the *final* classifier [7].

The most important difference that distinguishes our method from SVMs (and, by extension, from the general kernel discriminant family described above) is that

the points in our dyads are not typically located near the decision boundary, as is the case with support vectors. As a result, the final set of "basis vectors" used by the boosted strong classifier can be viewed as a *representative* subset of the data (i.e. those points needed for classification), whereas with SVMs the support vectors are simply the minimal number of training points needed to build (support) the decision boundary and are almost certainly not "typical" or high-likelihood members of either class.[2]

The *classification complexity* of a kernel-based classifier — the cost of classifying a test point — depends on the number of kernel function evaluations on which the classifier is based. In the case of SVMs, there is (usually) no direct way of controlling this number (the quadratic programming solution will automatically determine all positive Lagrange multipliers). In our boosted hypercut algorithm, however, the number of dyadic "basis vectors", and therefore of the required kernel evaluations, is determined by the number of iterations of the boosting algorithm and can therefore be controlled. Note that we are not referring here to the complexity of *training* classifiers here, only to their run-time computational cost.

## 2 Methodology

Consider a binary classification task where we are given a training set of vectors $T = \{\mathbf{x}_1, \ldots, \mathbf{x}_M\}$ where $\mathbf{x} \in R^N$, with corresponding labels $\{y_1, \ldots, y_M\}$ where $y \in \{-1, +1\}$. Let there be $M_p$ samples with label $+1$ and $M_n$ samples with label $-1$ so that $M = M_p + M_n$. Consider a simple linear hyperplane classifier defined by a discriminant function of the form

$$f(\mathbf{x}) = \langle \mathbf{w} \cdot \mathbf{x} \rangle + b \tag{4}$$

where $\text{sign}(f(\mathbf{x})) \in \{+1, -1\}$ gives the binary classification.

Under certain assumptions, Gaussianity in particular, the optimal hyperplane, specified by the projection $\mathbf{w}^*$ and bias $b^*$, is easily computed using standard statistical techniques based on class means and sample covariances for linear classifiers. However, in the absence of such assumptions, one must resort to searching for the optimal hyperplane. When searching for $\mathbf{w}^*$, an efficient strategy is to consider only hyperplanes whose surface normal is parallel to the line joining a dyad $(\mathbf{x}_i, \mathbf{x}_j)$:

$$\mathbf{w}_{ij} = \frac{\mathbf{x}_i - \mathbf{x}_j}{c}, \quad y_i \neq y_j, \quad i < j \tag{5}$$

where $y_i \neq y_j$ by definition, $i < j$ for uniqueness, and $c$ is a scale factor. The vector $\mathbf{w}_{ij}$ is parallel to the line segment connecting the points in a dyad. Setting $c = \|\mathbf{x}_i - \mathbf{x}_j\|$ makes $\mathbf{w}_{ij}$ a unit-norm direction vector.

The hypothesis space to be searched consists of $|\{\mathbf{w}_{ij}\}| = M_p M_n$ hypercuts, each having a free bias parameter $b_{ij}$ which is typically determined by minimizing the weighted classification error (as we shall see in the next section). Each hypothesis is then given by the sign of the discriminant as in (4):

$$h_{ij}(\mathbf{x}) = \text{sign}(\langle \mathbf{w}_{ij} \cdot \mathbf{x} \rangle + b_{ij}) \tag{6}$$

Let $\{h_{ij}\} = \{\mathbf{w}_{ij}, b_{ij}\}$ denote the complete set of hypercuts for a given training set. Strictly speaking, this set is uncountable since $b_{ij}$ is continuous and arbitrary. However, since we always select one bias parameter for each hypercut $\mathbf{w}_{ij}$, we do in fact end up with only $M_p M_n$ classifiers.

## 2.1 AdaBoost

The AdaBoost algorithm [4] provides a practical framework for combining a number of weak classifiers into a strong final classifier by means of linear combination and thresholding. AdaBoost works by maintaining over the training set an iteratively evolving distribution (weights) $D_t(i)$ based on the difficulty of classification (i.e. points which are harder to classify have greater weight). Consequently, a "weak" hypothesis $h(\mathbf{x}) : \mathbf{x} \to \{+1, -1\}$ will have classification error $\epsilon_t$ weighted by $D_t$. In our case, in each iteration $t$, we select from the complete set of $M_p M_n$ hypercuts $\{h_{ij}\}$ one which minimizes $\epsilon_t$. The data are then re-weighted based on their (mis)classification to obtain an updated distribution $D_{t+1}$.

The final classifier is a linear combination of the selected weak classifiers $h_t$ and has the form of a weighted "voting" scheme

$$H(\mathbf{x}) = \text{sign} \left( \sum_{i=1}^{T} \alpha_t h_t(\mathbf{x}) \right) \qquad (7)$$

where $\alpha_t = \frac{1}{2} \ln(\frac{1-\epsilon_t}{\epsilon_t})$. In [7] a framework was developed where $h_t(\mathbf{x})$ can be real-valued (as opposed to binary) and is interpreted as a "confidence-rated prediction." The sign of $h_t(\mathbf{x})$ is the predicted label while the magnitude $\mid h_t(\mathbf{x}) \mid$ is the confidence. For such real-valued classifiers we have

$$\alpha_t = \frac{1}{2} \ln \left( \frac{1+r_t}{1-r_t} \right) \qquad (8)$$

where the "correlation" $r_t = \sum_i D_t(i) \, y_i \, h_t(\mathbf{x_i})$ is inversely related to the error by $\epsilon_t = (1 - r_t)/2$.

## 2.2 Nonlinear Hypercuts

The logical extension beyond the boosted linear dyadic discriminants described in the previous section is that of nonlinear discriminants using positive definite *kernels* as suggested in [3] for use with SVMs. In the resulting "reproducing kernel Hilbert spaces", dot products between high-dimensional mappings $\Phi(\mathbf{x}) : \mathbf{X} \to \mathbf{F}$ are easily evaluated using Mercer kernels

$$k(\mathbf{x}, \mathbf{x}') = \langle \Phi(\mathbf{x}) \cdot \Phi(\mathbf{x}') \rangle. \qquad (9)$$

This has the desirable property that any algorithm based on dot products, e.g. our linear hypercut classifier (6), can first nonlinearly transform its inputs (using kernels) and implicitly perform dot-products in the transformed space. The pre-image of the linear hyperplane solution back in the input space is thus a nonlinear hypersurface.

Applying the above kernel property to the hypercut concept (5) we can rewrite it in nonlinear form by considering the linear hypercut in the transformed space $\mathbf{F}$ where the projection operator is

$$\mathbf{w}_{ij} = \Phi(\mathbf{x}_i) - \Phi(\mathbf{x}_j), \quad y_i \neq y_j, \quad i < j \qquad (10)$$

(we have absorbed the scale constant $c$ in (5) into $\mathbf{w}_{ij}$ for simplicity in this case).[3] Due to the implicit nature of the nonlinear mapping, we can not directly evaluate $\mathbf{w}_{ij}$. However, we only need its dot product with the transformed input vectors

$\Phi(\mathbf{x})$. Considering the linear discriminant (4) and substituting the above we obtain

$$f_{ij}(\mathbf{x}) \;=\; \langle (\Phi(\mathbf{x}_i) - \Phi(\mathbf{x}_j)) \cdot \Phi(\mathbf{x}) \rangle \;+\; b_{ij}, \tag{11}$$

which by applying the kernel property (9) is equivalent to

$$f_{ij}(\mathbf{x}) \;=\; k(\mathbf{x}, \mathbf{x}_i) - k(\mathbf{x}, \mathbf{x}_j) + \; b_{ij} \tag{12}$$

Note that $f_{ij}$ now represents a single dyadic term in the potential function introduced in (3). The binary-valued hypercut classifier is given by a simple thresholding

$$h_{ij}(\mathbf{x}) \;=\; \mathrm{sign}(f_{ij}(\mathbf{x})). \tag{13}$$

A "confidence-rated" classifier with output in the range $[-1, +1]$ can be obtained by passing $f_{ij}$ through a bipolar sigmoidal nonlinearity such as a hyperbolic tangent

$$h_{ij}(\mathbf{x}) \;=\; \tanh\left(\beta f_{ij}(\mathbf{x})\right) \tag{14}$$

where $\beta$ determines the "slope" of the sigmoid. We note that in order to obtain a continuous-valued hypercut classifier that suitably occupies the range $[-1, +1]$ it may be necessary to experiment and adjust both constants $c$ and $\beta$.

The final classifier constructed by AdaBoost, following (7), is given by

$$H(\mathbf{x}) = \mathrm{sign}\left( \sum_{t=1}^{T} \alpha_t \tanh\left(\beta\left[k(\mathbf{x}, \mathbf{x}_i^t) - k(\mathbf{x}, \mathbf{x}_j^t) + b_{ij}^t\right]\right)\right), \tag{15}$$

where we have superscripted the elements of $f_{ij}$ selected in iteration $t$ of boosting. Note that besides the monotonic sigmoid and offset transformation, this form is essentially a (nonlinear) equivalent of the dyadic potential function of (3).

If we assume, without loss of generality, that an equal number $N/2$ of $d$-dimensional training points is available from each class, defining $O(N^2)$ hypercuts. The values of $f_{ij}(\mathbf{x})$ for each hypercut and each training point (12) can be computed only once, typically in $O(d)$, and used in every iteration of the algorithm, making the setup cost for the algorithm $O(dN^3)$. Each iteration requires examination of all $f_{ij}(x_k)$ and takes $O(N^3)$. To summarize, the cost of learning a classifier with $K$ dyads is $O\left((d + K)N^3\right)$. It is important to note that both the setup step and the search for an optimal hypercut in each iteration are naturally parallelizable, leading to a reduction in time linear in the number of processors.

## 3   Experiments

Before applying our algorithm to standard benchmarks, we illustrate a simple 2D example of nonlinear boosted dyadic hypercuts on a "toy" problem. Consider a classification task on the dataset of 20 points (10 for each class) shown in Figure 1. The hypercuts algorithm (using Gaussian kernels) was able to separate the classes using two iterations (two cuts) as shown in Figure 1(a). Note how the dyads of training points (connected by dashed lines) define the discriminant boundary. For comparison, we used an SVM with Gaussian kernels on the same dataset, as shown in Figure 1(b). Although the SVM has a wider margin, the same would be expected from our algorithm with additional rounds of boosting.

The computational cost of classifying a point can be directly compared in terms of the number of required kernel evaluations in (2), which dominate the computation for high-dimensional data and kernels like Gaussians. For SVM, this is the number of support vectors. For hypercuts, this is the number of *distinct* training points

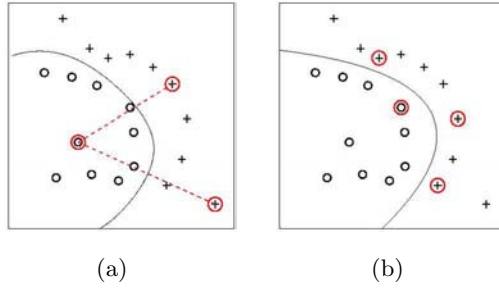

(a)          (b)

Figure 1: A toy problem: classification based on (a) hypercuts (2 dyads) (b) SVM (4 support vectors).

in the selected dyads. After $n$ rounds of boosting this number is bounded by $2n$, since a point can participate in multiple dyads. For instance, the SVM in Figure 1 requires 4 kernel evaluations, compared to 3 for the boosted hypercuts.

## 3.1 Experiments with real data sets

We evaluated the performance of the dyadic hypercuts algorithm on a number of real-world data sets from the UCI repository [2], and compared the performance to that of two established classification methods: SVM with Gaussian RBF kernel and $k$-Nearest Neighbor ($k$-NN). We chose sets large enough for reasonable training/validation/test partitioning, and that represent binary (or easily converted to binary) classification problems.

| Dataset | $N$ | $d$ | $k$-NN | SVM | #SV | Hypercuts | #k.ev. |
|---|---|---|---|---|---|---|---|
| HEART | 90 | 13 | .196 ±.042 | .202 ±.038 | 62 ±10 | .202 ±.030 | 50 ±12 |
| IONOSPHERE | 120 | 34 | .168 ±.024 | .064 ±.018 | 73 ±7 | .083 ±.022 | 63 ±7 |
| WBC | 200 | 9 | .034 ±.011 | .032 ±.008 | 50 ±26 | .028 ±.007 | 30 ±12 |
| WPBC | 65 | 32 | .250 ±.024 | .243 ±.006 | **63** ±3 | .253 ±.025 | **41** ±5 |
| WDBC | 190 | 30 | .044 ±.015 | .035 ±.013 | 67 ±15 | .038 ±.014 | 47 ±12 |
| WINE | 60 | 13 | .053 ±.030 | .032 ±.022 | **40** ±9 | .040 ±.026 | **23** ±4 |
| SPAM | 150 | 57 | .159 ±.025 | .123 ±.016 | **101** ±8 | .116 ±.019 | **73** ±15 |
| SONAR | 70 | 60 | .227 ±.041 | .226 ±.037 | **66** ±3 | .202 ±.045 | **52** ±5 |
| PIMA | 200 | 8 | .267 ±.024 | .244 ±.014 | 129 ±7 | .260 ±.017 | 110 ±16 |

Table 1: The results of the experiments described in Section 3.1. $N$ is the size of the training set, $d$ the dimension, #SV the number of support vectors for the SVM, and #k.ev. the number of kernel evaluations required by a boosted hypercuts classifier. Means and standard deviations in 30 trials are reported for each data set. WBC,WPBC,WDBC are Wisconsin Breast Cancer, Prognosis and Diagnosis data sets, respectively.

In each experiment, the data set was randomly partitioned into training, validation and test sets of similar sizes. The validation set was used to "tune" the parameters of each of the classifiers ($k$ for $k$-NN, $\sigma$ for RBF kernels of SVMs and hypercuts), by choosing from a suitable range the parameter value with lowest error on the validation set. Each of the three classifiers was then trained with the chosen parameter on the training set, and tested on the test set.

For each data data set the above experiment was repeated 30 times. The columns of Table 1, left to right, show the following, with means and standard deviations over the 30 trials for each dataset: size of the training set, dimension, the test error

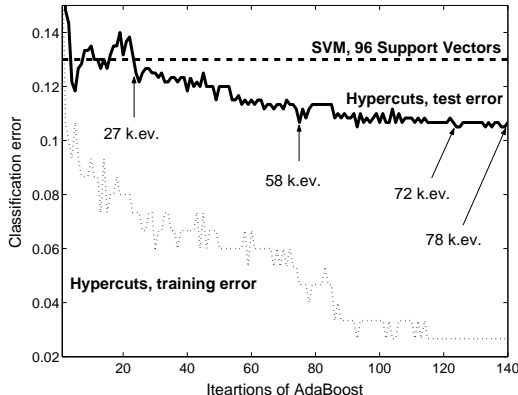

Figure 2: An example of the progress of training (dotted line) and test (solid line) error in a run of hypercuts algorithm with RBF kernel on SPAM data. The number of kernel evaluations in the combined classifier is shown for indicated points in the run. The dashed line shows the test error of the SVM with RBF kernel.

| Dataset | 10% | 25% | 50% |
|---------|-----|-----|-----|
| HEART | .202 | .200 | .197 |
| ION. | .178 | .113 | .094 |
| WBC | .028 | .028 | . 028 |
| WPBC | .302 | .269 | .266 |
| WDBC | .365 | .384 | .383 |
| WINE | .064 | .051 | .043 |
| SPAM | .142 | .124 | .117 |
| SONAR | .248 | .233 | .214 |
| PIMA | .269 | .268 | .263 |

Table 2: Test error as a function of number of kernel evaluations allowed by the user; the percentage values are relative to the number of SVs in each experiment. Averaged over 30 trials for each data set.

of $k$-NN, the test error of SVM, the number of support vectors, the test error of hypercuts, and the number of kernel evaluations in the final hypercuts classifier.

The size of the hypercuts classifier can be controlled via the number of AdaBoost iterations, thus affecting the accuracy of the classifier. In our experiments boosting was stopped after a prolonged plateau in the training error was observed; in some cases, further continuation of boosting could lead to better results.

## 3.2   Discussion

The most important conclusion from these empirical results is that for all data sets, the RBF boosted dyadic hypercuts achieve test performance statistically equivalent to that of SVMs[4], and usually better than that of $k$-NN classifiers, while the complexity of the trained classifier is typically lower (in some cases, which appear in bold in Table 1, the difference in complexity is significant).

In addition, our experiments demonstrate the trade-off between the complexity and accuracy of the hypercuts. Figure 2 shows an example run of hypercuts algorithm on SPAM data set, with 150 training points. After 24 iterations, the test error of the final classifier becomes consistently lower than that of SVM trained on the same training set, which found 96 support vectors. At that point the classifier requires 27 kernel evaluations (about 28% of the number of SVs). The following 115 iterations achieve further improvement of only 1.8% in test error, while increasing the required number of kernel evaluations to 78. Here the automatic criterion stopped the AdaBoost after no significant improvement in training error was observed for 25 iterations. But the user can instead specify the desired bound on the complexity of the classifier. Table 2 shows the behavior of test error as a function of the number of kernel evaluations by the classifier, averaged over all 30 trials. For some data sets, e.g. HEART and WBC, the hypercuts classifier with only 10% of the number of kernel evaluations in an SVM already achieves comparable test error.

# 4   Conclusions

The contribution of this paper is two-fold. First, we proposed a family of simple discriminants (hypercuts), based on pairs of training points from opposite classes (dyads), and extended this family using a nonlinear mapping with Mercer-type kernels. Second, we have designed a greedy selection algorithm based on boosting with confidence-rated (real-valued) hypercut classifiers with continuous output in the interval [-1,1].

This is a new kernel based approach to classification. We have shown that this approach performs on par with SVMs, without having to solve large QP problems. In contrast, our algorithm allows the user to trade off the classifier's computational complexity for its accuracy, and benefits from AdaBoost's exponential error convergence and the assurance of asymptotic margin maximization.

The generalization performance of our algorithm was evaluated on a number of data sets from the UCI repository, and demonstrated to be comparable to that of established state-of-the-art algorithms (SVMs, $k$-NN), often with reduced classification time and reduced classifier size. We emphasize this performance advantage, since in practical applications it is often desirable to minimize complexity even at the cost of increased training time.

We are currently looking into optimal strategies for sampling the hypothesis space ($M_p M_n$ possible hypercuts) based on the distribution $D_t(i)$ and forming hypercuts that are not necessarily based on training samples but rather, for example, on cluster centroids or other points derived from the input distribution. This has the potential to dramatically reduce the computational cost of learning in the boosted hypercuts algorithm, thus making it even more attractive for a practitioner.

## Footnotes

[1]The physical analogy here is to the linear superposition of electrostatic charges of strength $\alpha_i$, polarity $y_i$ and location $\mathbf{x_i}$ with distance defined by the kernel $K$.

[2]Although unrelated to our technique, the Relevance Vector machine [8] is another kernel learning algorithm that tends to produce "prototypical" basis vectors in the interior as opposed to the boundary of the distributions.

[3]Since the optimal projection $\mathbf{w}_{ij}^*$ must lie in the span of $\{\Phi(\mathbf{x}_i)\}$, we should restrict the search for an optimal hyperplane accordingly, e.g. by considering pair-wise hypercuts.

[4]i.e. the difference of the means is within one standard deviation from both sides

# References

[1] M. A. Aizerman, E. M. Braverman, and L. I. Rozonoer. Theoretical foundations of the potential function method in pattern recognition learning. *Automation and Remote Control*, 25:821–837, 1964.

[2] C. L. Blake and C. J. Merz. UCI repository of machine learning databases. [http://www.ics.uci.edu/~mlearn/MLRepository.html], 1998.

[3] B. E. Boser, I. M. Guyon, and V. N. Vapnik. A training algorithm for optimal margin classifiers. In D. Haussler, editor, *Proc. 5th Annual ACM Workshop on Computational Learning Theory*, pages 144–152. ACM Press, 1992.

[4] Y. Freund and R. E. Schapire. A decision-theoretic generalization of on-line learning and an application to boosting. *Journal of Computer and System Sciences*, 55(1):119–139, 1995.

[5] T. Jaakkola and D. Haussler. Probabilistic kernel regression models. In D. Heckerman and J. Whittaker, editors, *Proc. of 7th International Workshop on AI and Statistics*. Morgan Kaufman, 1999.

[6] P. McCallugh and J. Nelder. *Generalized Linear Models*. Chapman and Hall, London, 1983.

[7] Robert E. Schapire and Yoram Singer. Improved boosting algorithms using confidence-rated predictions. In *Proc. of 11th Annual Conf. on Computational Learning Theory*, pages 80–91, 1998.

[8] M. E. Tipping. The Relevance Vector Machine. In *Advances in Neural Information Processing Systems 12*, pages 652–658. MIT Press, 2000.

[9] V. Vapnik. *The Nature of Statistical Learning Theory*. Springer, 1995.
